# Gradient and Hamiltonian Dynamics Applied to Learning in Neural Networks

James W. Howse      Chaouki T. Abdallah      Gregory L. Heileman

Department of Electrical and Computer Engineering
University of New Mexico
Albuquerque, NM 87131

## Abstract

The process of machine learning can be considered in two stages: model selection and parameter estimation. In this paper a technique is presented for constructing dynamical systems with desired qualitative properties. The approach is based on the fact that an $n$-dimensional nonlinear dynamical system can be decomposed into one gradient and $(n - 1)$ Hamiltonian systems. Thus, the model selection stage consists of choosing the gradient and Hamiltonian portions appropriately so that a certain behavior is obtainable. To estimate the parameters, a stably convergent learning rule is presented. This algorithm has been proven to converge to the desired system trajectory for all initial conditions and system inputs. This technique can be used to design neural network models which are guaranteed to solve the trajectory learning problem.

## 1   Introduction

A fundamental problem in mathematical systems theory is the identification of dynamical systems. System identification is a dynamic analogue of the functional approximation problem. A set of input-output pairs $\{u(t), y(t)\}$ is given over some time interval $t \in [\mathcal{T}_i, \mathcal{T}_f]$. The problem is to find a model which for the given input sequence returns an approximation of the given output sequence. Broadly speaking, solving an identification problem involves two steps. The first is choosing a class of identification models which are capable of emulating the behavior of the actual system. The second is selecting a method to determine which member of this class of models best emulates the actual system. In this paper we present a class of nonlinear models and a learning algorithm for these models which are guaranteed to learn the trajectories of an example system. Algorithms to learn given trajectories of a continuous time system have been proposed in [6], [8], and [7] to name only a few. To our knowledge, no one has ever proven that the error between the learned and desired trajectories vanishes for any of these algorithms. In our trajectory learning system this error is guaranteed to vanish. Our models extend the work in [1] by showing that Cohen's systems are one instance of the class of models generated by decomposing the dynamics into a component normal to some surface and a set of components tangent to the same surface. Conceptually this formalism can be used to design dynamical systems with a variety of desired qualitative properties. Furthermore, we propose a provably convergent learning algorithm which allows the parameters of Cohen's models to be learned from examples rather than being programmed in advance. The algorithm is

convergent in the sense that the error between the model trajectories and the desired trajectories is guaranteed to vanish. This learning procedure is related to one discussed in [5] for use in linear system identification.

## 2    Constructing the Model

First some terminology will be defined. For a system of $n$ first order ordinary differential equations, the *phase space* of the system is the $n$-dimensional space of all state components. A solution *trajectory* is a curve in phase space described by the differential equations for one specific starting point. At every point on a trajectory there exists a tangent vector. The space of all such tangent vectors for all possible solution trajectories constitutes the *vector field* for this system of differential equations.

The trajectory learning models in this paper are systems of first order ordinary differential equations. The form of these equations will be obtained by considering the system dynamics as motion relative to some surface. At each point in the state space an arbitrary system trajectory will be decomposed into a component normal to this surface and a set of components tangent to this surface. This approach was suggested to us by the results in [4], where it is shown that an arbitrary $n$-dimensional vector field can be decomposed locally into the sum of one gradient vector field and $(n-1)$ Hamiltonian vector fields. The concept of a potential function will be used to define these surfaces. A *potential function* $V(x)$ is any scalar valued function of the system states $x = [x_1, x_2, \ldots, x_n]^\dagger$ which is at least twice continuously differentiable (i.e. $V(x) \in C^r : r \geq 2$). The operation $[\cdot]^\dagger$ denotes the transpose of the vector. If there are $n$ components in the system state, the function $V(x)$, when plotted with respect all of the state components, defines a surface in an $(n+1)$-dimensional space. There are two curves passing through every point on this potential surface which are of interest in this discussion, they are illustrated in Figure 1(a). The dashed curve is

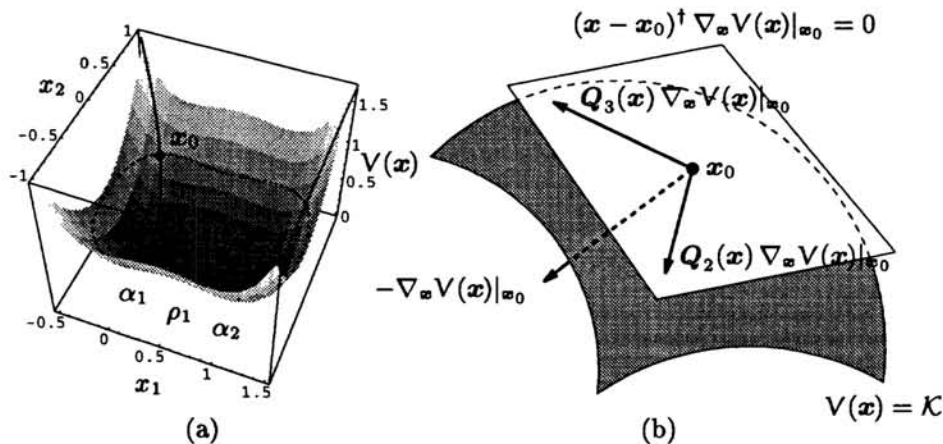

**Figure 1:** (a) The potential function $V(x) = x_1^2 (x_1 - 1)^2 + x_2^2$ plotted versus its two dependent variables $x_1$ and $x_2$. The dashed curve is called a level surface and is given by $V(x) = 0.5$. The solid curve follows the path of steepest descent through $x_0$. (b) The partitioning of a 3-dimensional vector field at the point $x_0$ into a 1-dimensional portion which is normal to the surface $V(x) = \mathcal{K}$ and a 2-dimensional portion which is tangent to $V(x) = \mathcal{K}$. The vector $-\nabla_x V(x)|_{x_0}$ is the normal vector to the surface $V(x) = \mathcal{K}$ at the point $x_0$. The plane $(x - x_0)^\dagger \nabla_x V(x)|_{x_0} = 0$ contains *all* of the vectors which are tangent to $V(x) = \mathcal{K}$ at $x_0$. Two linearly independent vectors are needed to form a basis for this tangent space, the pair $Q_2(x) \nabla_x V(x)|_{x_0}$ and $Q_3(x) \nabla_x V(x)|_{x_0}$ that are shown are just one possibility.

referred to as a *level surface*, it is a surface along which $V(x) = \mathcal{K}$ for some constant $\mathcal{K}$. Note that in general this level surface is an $n$-dimensional object. The solid curve

moves downhill along $V(x)$ following the path of steepest descent through the point $x_0$. The vector which is tangent to this curve at $x_0$ is normal to the level surface at $x_0$. The system dynamics will be designed as motion relative to the level surfaces of $V(x)$. The results in [4] require $n$ different local potential functions to achieve arbitrary dynamics. However, the results in [1] suggest that a considerable number of dynamical systems can be achieved using only a single global potential function.

A system which is capable of traversing any downhill path along a given potential surface $V(x)$, can be constructed by decomposing each element of the vector field into a vector normal to the level surface of $V(x)$ which passes through each point and a set of vectors tangent to the level surface of $V(x)$ which passes through the same point. So the potential function $V(x)$ is used to partition the $n$-dimensional phase space into two subspaces. The first contains a vector field normal to some level surface $V(x) = \mathcal{K}$ for $\mathcal{K} \in \mathbb{R}$, while the second subspace holds a vector field tangent to $V(x) = \mathcal{K}$. The subspace containing all possible normal vectors to the $n$-dimensional level surface at a given point, has dimension one. This is equivalent to the statement that every point on a smooth surface has a unique normal vector. Similarly, the subspace containing all possible tangent vectors to the level surface at a given point has dimension $(n-1)$. An example of this partition in the case of a 3-dimensional system is shown in Figure 1(b). Since the space of all tangent vectors at each point on a level surface is $(n-1)$-dimensional, $(n-1)$ linearly independent vectors are required to form a basis for this space.

Mathematically, there is a straightforward way to construct dynamical systems which either move downhill along $V(x)$ or remain at a constant height on $V(x)$. In this paper, dynamical systems which always move downhill along some potential surface are called *gradient-like systems*. These systems are defined by differential equations of the form

$$\dot{x} = -P(x)\,\nabla_x V(x), \tag{1}$$

where $P(x)$ is a matrix function which is symmetric (i.e. $P^\dagger = P$) and positive definite at every point $x$, and where $\nabla_x V(x) = [\frac{\partial V}{\partial x_1}, \frac{\partial V}{\partial x_2}, \ldots, \frac{\partial V}{\partial x_n}]^\dagger$. These systems are similar to the gradient flows discussed in [2]. The trajectories of the system formed by Equation (1) always move downhill along the potential surface defined by $V(x)$. This can be shown by taking the time derivative of $V(x)$ which is $\dot{V}(x) = -[\nabla_x V(x)]^\dagger P(x) [\nabla_x V(x)] \leq 0$. Because $P(x)$ is positive definite, $\dot{V}(x)$ can only be zero where $\nabla_x V(x) = 0$, elsewhere $\dot{V}(x)$ is negative. This means that the trajectories of Equation (1) always move toward a level surface of $V(x)$ formed by "slicing" $V(x)$ at a lower height, as pointed out in [2]. It is also easy to design systems which remain at a constant height on $V(x)$. Such systems will be denoted *Hamiltonian-like systems*. They are specified by the equation

$$\dot{x} = Q(x)\,\nabla_x V(x), \tag{2}$$

where $Q(x)$ is a matrix function which is skew-symmetric (i.e. $Q^\dagger = -Q$) at every point $x$. These systems are similar to the Hamiltonian systems defined in [2]. The elements of the vector field defined by Equation (2) are always tangent to some level surface of $V(x)$. Hence the trajectories of this system remain at a constant height on the potential surface given by $V(x)$. Again this is indicated by the time derivative of $V(x)$, which in this case is $\dot{V}(x) = [\nabla_x V(x)]^\dagger Q(x) [\nabla_x V(x)] = 0$. This indicates that the trajectories of Equation (2) always remain on the level surface on which the system starts. So a model which can follow an arbitrary downhill path along the potential surface $V(x)$ can be designed by combining the dynamics of Equations (1) and (2). The dynamics in the subspace normal to the level surfaces of $V(x)$ can be

defined using one equation of the form in Equation (1). Similarly the dynamics in the subspace tangent to the level surfaces of $V(x)$ can be defined using $(n-1)$ equations of the form in Equation (2). Hence the total dynamics for the model are

$$\dot{x} = -P(x)\,\nabla_x V(x) + \sum_{i=2}^{n} Q_i(x)\,\nabla_x V(x). \tag{3}$$

For this model the number and location of equilibria is determined by the function $V(x)$, while the manner in which the equilibria are approached is determined by the matrices $P(x)$ and $Q_i(x)$.

If the potential function $V(x)$ is bounded below (i.e. $V(x) > \mathcal{B}_l \; \forall \; x \in \mathbb{R}^n$, where $\mathcal{B}_l$ is a constant), eventually increasing (i.e. $\lim_{\|x\| \to \infty} V(x) \to \infty$) , and has only a finite number of isolated local maxima and minima (i.e. in some neighborhood of every point where $\nabla_x V(x) = 0$ there are no other points where the gradient vanishes), then the system in Equation (3) satisfies the conditions of Theorem 10 in [1]. Therefore the system will converge to one of the points where $\nabla_x V(x) = 0$, called the *critical points* of $V(x)$, for all initial conditions. Note that this system is capable of all downhill trajectories along the potential surface only if the $(n-1)$ vectors $Q_i(x)\,\nabla_x V(x) \; \forall \; i = 2,\dots,n$ are linearly independent at every point $x$. It is shown in [1] that the potential function

$$V(x) = C \int_{\mathcal{X}_1}^{x_1} \mathcal{L}_1(\gamma)\, d\gamma + \sum_{i=2}^{n} \left[ \frac{1}{2} (x_i - \mathcal{L}_i(x_1))^2 + \frac{1}{2} \int_{\mathcal{X}_i}^{x_1} \mathcal{L}_1(\gamma)\, [\mathcal{L}_i'(\gamma)]^2 \, d\gamma \right] \tag{4}$$

satisfies these three criteria. In this equation $\mathcal{L}_i(x_1) \; \forall \; i = 1,\dots,n$ are interpolation polynomials, $C$ is a real positive constant, $\mathcal{X}_i \; \forall \; i = 1,\dots,n$ are real constants chosen so that the integrals are positive valued, and $\mathcal{L}_i'(x_1) \equiv \frac{d\mathcal{L}_i}{dx_1}$.

## 3  The Learning Rule

In Equation (3) the number and location of equilibria can be controlled using the potential function $V(x)$, while the manner in which the equilibria are approached can be controlled with the matrices $P(x)$ and $Q_i(x)$. If it is assumed that the locations of the equilibria are known, then a potential function which has local minima and maxima at these points can be constructed using Equation (4). The problem of trajectory learning is thereby reduced to the problem of parameterizing the matrices $P(x)$ and $Q_i(x)$ and finding the parameter values which cause this model to best emulate the actual system. If the elements $P(x)$ and $Q_i(x)$ are correctly chosen, then a learning rule can be designed which makes the model dynamics converge to that of the actual system. Assume that the dynamics given by Equation (3) are a parameterized model of the actual dynamics. Using this model and samples of the actual system states, an estimator for states of the actual system can be designed. The behavior of the model is altered by changing its parameters, so a parameter estimator must also be constructed. The following theorem provides a form for both the state and parameter estimators which guarantees convergence to a set of parameters for which the error between the estimated and target trajectories vanishes.

**Theorem 3.1.** *Given the model system*

$$\dot{x} = \sum_{i=1}^{k} A_i\, f_i(x) + B\, g(u) \tag{5}$$

*where $A_i \in \mathbb{R}^{n \times n}$ and $B \in \mathbb{R}^{n \times m}$ are unknown, and $f_i(\cdot)$ and $g(\cdot)$ are known smooth functions such that the system has bounded solutions for bounded inputs $u(t)$. Choose*

*a state estimator of the form*

$$\dot{\hat{x}} = \mathcal{R}_s \left( \hat{x} - x \right) + \sum_{i=1}^{k} \hat{A}_i \, f_i(x) + \hat{B} \, g(u) \tag{6}$$

*where $\mathcal{R}_s$ is an $(n \times n)$ matrix of real constants whose eigenvalues must all be in the left half plane, and $\hat{A}_i$ and $\hat{B}$ are the estimates of the actual parameters. Choose parameter estimators of the form*

$$\dot{\hat{A}}_i = -\mathcal{R}_p \left( \hat{x} - x \right) \left[ f_i(x) \right]^\dagger \ \forall \ i = 1, \dots, k$$

$$\dot{\hat{B}} = -\mathcal{R}_p \left( \hat{x} - x \right) \left[ g(u) \right]^\dagger \tag{7}$$

*where $\mathcal{R}_p$ is an $(n \times n)$ matrix of real constants which is symmetric and positive definite, and $(\hat{x} - x) \left[ \cdot \right]^\dagger$ denotes an outer product. For these choices of state and parameter estimators $\lim_{t \to \infty} (\hat{x}(t) - x(t)) = 0$ for all initial conditions. Furthermore, this remains true if any of the elements of $\hat{A}_i$ or $\hat{B}$ are set to 0, or if any of these matrices are restricted to being symmetric or skew-symmetric.*

The proof of this theorem appears in [3]. Note that convergence of the parameter estimates to the actual parameter values is not guaranteed by this theorem. The model dynamics in Equation (3) can be cast in the form of Equation (5) by choosing each element of $P(x)$ and $Q_i(x)$ to have the form

$$P_{rs} = \sum_{j=1}^{n} \sum_{k=0}^{l-1} \xi_{rsjk} \, \vartheta_k(x_j) \quad \text{and} \quad Q_{rs} = \sum_{j=1}^{n} \sum_{k=0}^{l-1} \lambda_{rsjk} \, \varrho_k(x_j), \tag{8}$$

where $\{\vartheta_0(x_j), \vartheta_1(x_j), \dots, \vartheta_{l-1}(x_j)\}$ and $\{\varrho_0(x_j), \varrho_1(x_j), \dots, \varrho_{l-1}(x_j)\}$ are a set of $l$ orthogonal polynomials which depend on the state $x_j$. There is a set of such polynomials for every state $x_j$, $j = 1, 2, \dots, n$. The constants $\xi_{rsjk}$ and $\lambda_{rsjk}$ determine the contribution of the $k$th polynomial which depends on the $j$th state to the value of $P_{rs}$ and $Q_{rs}$ respectively. In this case the dynamics in Equation (3) become

$$\dot{x} = \sum_{j=1}^{n} \sum_{k=0}^{l-1} \left\{ \Xi_{jk} \left[ \vartheta_k(x_j) \, \nabla_x V(x) \right] + \sum_{i=2}^{n} \Lambda_{ijk} \left[ \varrho_{ik}(x_j) \, \nabla_x V(x) \right] \right\} + \Upsilon \, g(u(t)) \tag{9}$$

where $\Xi_{jk}$ is the $(n \times n)$ matrix of all values $\xi_{rsjk}$ which have the same value of $j$ and $k$. Likewise $\Lambda_{ijk}$ is the $(n \times n)$ matrix of all values $\lambda_{rsjk}$, having the same value of $j$ and $k$, which are associated with the $i$th matrix $Q_i(x)$. This system has $m$ inputs, which may explicitly depend on time, that are represented by the $m$-element vector function $u(t)$. The $m$-element vector function $g(\cdot)$ is a smooth, possibly nonlinear, transformation of the input function. The matrix $\Upsilon$ is an $(n \times m)$ parameter matrix which determines how much of input $s \in \{1, \dots, m\}$ effects state $r \in \{1, \dots, n\}$. Appropriate state and parameter estimators can be designed based on Equations (6) and (7) respectively.

## 4   Simulation Results

Now an example is presented in which the parameters of the model in Equation (9) are trained, using the learning rule in Equations (6) and (7), on one input signal and then are tested on a different input signal. The actual system has three equilibrium points, two stable points located at $(1, 3)$ and $(3, 5)$, and a saddle point located at $(2 - \frac{\sqrt{3}}{3}, 4 + \frac{\sqrt{3}}{3})$. In this example the dynamics of both the actual system and the model are given by

$$\begin{pmatrix} \dot{x}_1 \\ \dot{x}_2 \end{pmatrix} = \begin{pmatrix} \mathcal{P}_1 + \mathcal{P}_2 \, x_1^2 + \mathcal{P}_3 \, x_2^2 & 0 \\ 0 & \mathcal{P}_4 + \mathcal{P}_5 \, x_1^2 + \mathcal{P}_6 \, x_2^2 \end{pmatrix} \begin{pmatrix} \frac{\partial V}{\partial x_1} \\ \frac{\partial V}{\partial x_2} \end{pmatrix} + \begin{pmatrix} 0 & -\{\mathcal{P}_7 + \mathcal{P}_8 \, x_1 + \mathcal{P}_9 \, x_2\} \\ \mathcal{P}_7 + \mathcal{P}_8 \, x_1 + \mathcal{P}_9 \, x_2 & 0 \end{pmatrix} \begin{pmatrix} \frac{\partial V}{\partial x_1} \\ \frac{\partial V}{\partial x_2} \end{pmatrix} + \begin{pmatrix} \mathcal{P}_{10} \\ 0 \end{pmatrix} u(t) \tag{10}$$

where $V(x)$ is defined in Equation (4) and $u(t)$ is a time varying input. For the actual system the parameter values were $\mathcal{P}_1 = \mathcal{P}_4 = -4$, $\mathcal{P}_2 = \mathcal{P}_5 = -2$, $\mathcal{P}_3 = \mathcal{P}_6 = -1$, $\mathcal{P}_7 = 1$, $\mathcal{P}_8 = 3$, $\mathcal{P}_9 = 5$, and $\mathcal{P}_{10} = 1$. In the model the 10 elements $\mathcal{P}_i$ are treated as the unknown parameters which must be learned. Note that the first matrix function is positive definite if the parameters $\mathcal{P}_1$–$\mathcal{P}_6$ are all negative valued. The second matrix function is skew-symmetric for all values of $\mathcal{P}_7$–$\mathcal{P}_9$. The two input signals used for training and testing were $u_1 = 10000\,(\sin\frac{1}{3}1000\,t + \sin\frac{2}{3}1000\,t)$ and $u_2 = 5000\sin1000\,t$. The phase space responses of the actual system to the inputs $u_1$ and $u_2$ are shown by the solid curves in Figures 3(b) and 3(a) respectively. Notice that both of these inputs produce a periodic attractor in the phase space of Equation (10). In order to evaluate the effectiveness of the learning algorithm the Euclidean distance between the actual and learned state and parameter values was computed and plotted versus time. The results are shown in Figure 2. Figure 2(a) shows these statistics when

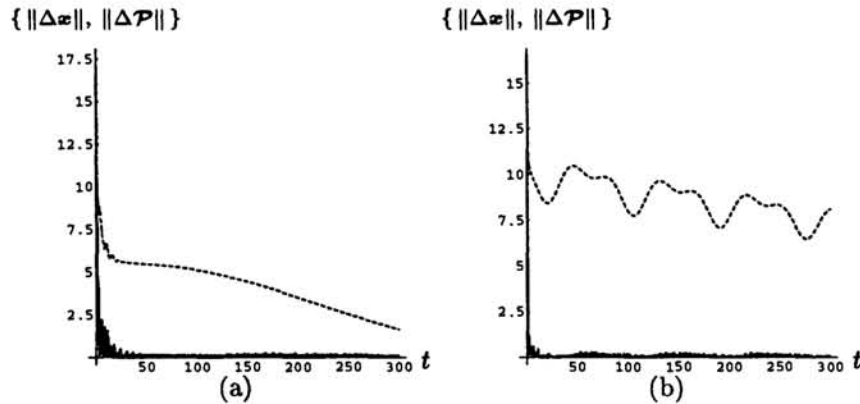

Figure 2:  (a) The state and parameter errors for training using input signal $u_1$. The solid curve is the Euclidean distance between the state estimates and the actual states as a function of time. The dashed curve shows the distance between the estimated and actual parameter values versus time.

(b) The state and parameter errors for training using input signal $u_2$.

training with input $u_1$, while Figure 2(b) shows the same statistics for input $u_2$. The solid curves are the Euclidean distance between the learned and actual system states, and the dashed curves are the distance between the learned and actual parameter values. These statistics have two noteworthy features. First, the error between the learned and desired states quickly converges to very small values, regardless of how well the actual parameters are learned. This result was guaranteed by Theorem 3.1. Second, the final error between the learned and desired parameters is much lower when the system is trained with input $u_1$. Intuitively this is because input $u_1$ excites more frequency modes of the system than input $u_2$. Recall that in a nonlinear system the frequency modes excited by a given input do not depend solely on the input because the system can generate frequencies not present in the input. The quality of the learned parameters can be qualitatively judged by comparing the phase plots using the learned and actual parameters for each input, as shown in Figure 3. In Figure 3(a) the system was trained using input $u_1$ and tested with input $u_2$, while in Figure 3(b) the situation was reversed. The solid curves are the system response using the actual parameter values, and the dashed curves are the response for the learned parameters. The Euclidean distance between the target and test trajectories in Figure 3(a) is in the range $(0, 0.64)$ with a mean distance of 0.21 and a standard deviation of 0.14. The distance between the the target and test trajectories in Figure 3(b) is in the range $(0, 4.53)$ with a mean distance of 0.98 and a standard deviation of 1.35. Qualitatively, both sets of learned parameters give an accurate response for non-training inputs.

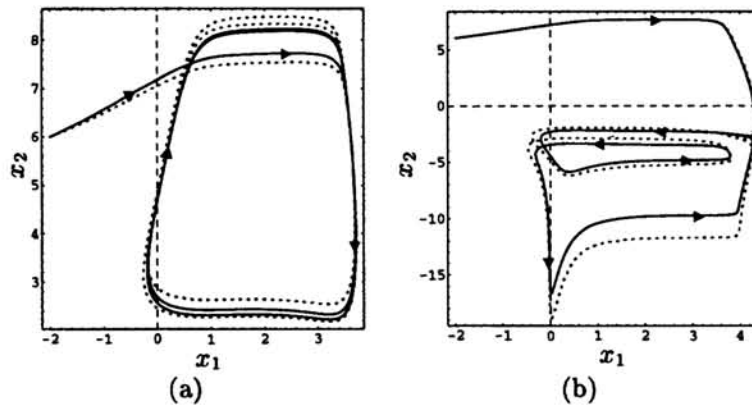

(a) &emsp;&emsp;&emsp;&emsp;&emsp;&emsp;&emsp;&emsp;&emsp;&emsp; (b)

**Figure 3:** (a) A phase plot of the system response when trained with input $u_1$ and tested with input $u_2$. The solid line is the response to the test input using the actual parameters. The dotted line is the system response using the learned parameters. (b) A phase plot of the system response when trained with input $u_2$ and tested with input $u_1$.

Note that even when the error between the learned and actual parameters is large, the periodic attractor resulting from the learned parameters appears to have the same "shape" as that for the actual parameters.

## 5 &emsp; Conclusion

We have presented a conceptual framework for designing dynamical systems with specific qualitative properties by decomposing the dynamics into a component normal to some surface and a set of components tangent to the same surface. We have presented a specific instance of this class of systems which converges to one of a finite number of equilibrium points. By parameterizing these systems, the manner in which these equilibrium points are approached can be fitted to an arbitrary data set. We present a learning algorithm to estimate these parameters which is guaranteed to converge to a set of parameter values for which the error between the learned and desired trajectories vanishes.

## Acknowledgments

This research was supported by a grant from Boeing Computer Services under Contract W–300445. The authors would like to thank Vangelis Coutsias, Tom Caudell, and Bill Horne for stimulating discussions and insightful suggestions.

## References

[1] M.A. Cohen. The construction of arbitrary stable dynamics in nonlinear neural networks. *Neural Networks*, 5(1):83–103, 1992.

[2] M.W. Hirsch and S. Smale. *Differential equations, dynamical systems, and linear algebra*, volume 60 of *Pure and Applied Mathematics*. Academic Press, Inc., San Diego, CA, 1974.

[3] J.W. Howse, C.T. Abdallah, and G.L. Heileman. A gradient-hamiltonian decomposition for designing and learning dynamical systems. Submitted to *Neural Computation*, 1995.

[4] R.V. Mendes and J.T. Duarte. Decomposition of vector fields and mixed dynamics. *Journal of Mathematical Physics*, 22(7):1420–1422, 1981.

[5] K.S. Narendra and A.M. Annaswamy. *Stable adaptive systems*. Prentice-Hall, Inc., Englewood Cliffs, NJ, 1989.

[6] B.A. Pearlmutter. Learning state space trajectories in recurrent neural networks. *Neural Computation*, 1(2):263–269, 1989.

[7] D. Saad. Training recurrent neural networks via trajectory modification. *Complex Systems*, 6(2):213–236, 1992.

[8] M.-A. Sato. A real time learning algorithm for recurrent analog neural networks. *Biological Cybernetics*, 62(2):237–241, 1990.
